# What Does the Hippocampus Compute?: A Precis of the 1993 NIPS Workshop

**Mark A. Gluck**
Center for Molecular and Behavioral Neuroscience
Rutgers University
Newark, NJ 07102
*gluck@pavlov.rutgers.edu*

Computational models of the hippocampal-region provide an important method for understanding the functional role of this brain system in learning and memory. The presentations in this workshop focused on how modeling can lead to a unified understanding of the interplay among hippocampal physiology, anatomy, and behavior. Several approaches were presented. One approach can be characterized as "top-down" analyses of the neuropsychology of memory, drawing upon brain-lesion studies in animals and humans. Other models take a "bottom-up" approach, seeking to infer emergent computational and functional properties from detailed analyses of circuit connectivity and physiology (see Gluck & Granger, 1993, for a review). Among the issues discussed were: (1) integration of physiological and behavioral theories of hippocampal function, (2) similarities and differences between animal and human studies, (3) representational vs. temporal properties of hippocampal-dependent behaviors, (4) rapid vs. incremental learning, (5) multiple vs. unitary memory systems, (5) spatial navigation and memory, and (6) hippocampal interaction with other brain systems.

Jay McClelland, of Carnegie-Mellon University, presented one example of a top-down approach to theory development in his talk, "Complementary roles of neocortex and hippocampus in learning and memory" McClelland reviewed findings indicating that the hippocampus appears necessary for the initial acquisition of some forms of memory, but that ultimately all forms of memory are stored independently of the hippocampal system. Consolidation in the neocortex appears to occur over an extended period -- in humans the process appears to extend over several years. McClelland suggested that the cortex uses interleaved learning to extract the structure of events and experiences while the hippocampus provides a special system for the rapid initial storage of traces of specific events and experiences in a form that minimizes mutual interference between memory traces. According to this view, the hippocampus is necessary to avoid the catastrophic

interference that would result if memories were stored directly in the neocortex. Consolidation is slow to allow the gradual integration of new knowledge via continuing interleaved learning (McClelland, 1994/in press).

In another example of top-down modeling, Mark Gluck of Rutgers University discussed "Stimulus representation and hippocampal function in animal and human learning." He described a computational account of hippocampal-region function in classical conditioning (Gluck & Myers, 1993; Myers & Gluck, 1994). In this model, the hippocampal region constructs new stimulus representations biased by two opponent constraints: first, to differentiate representations of stimuli which predict different future events, and second, to compress together representations of co-occurring or redundant stimuli. This theory accurately describe the role of the hippocampal region in a wide range of conditioning paradigms. Gluck also presented an extension of this theory which suggests that stimulus compression may arise from the operation of circuitry in the superficial layers of entorhinal cortex, whereas stimulus differentiation may arise from the operation of constituent circuits of the hippocampal formation.

Discussing more physiologically-motivated "bottom-up" research, Michael Hasselmo, of Harvard University, talked about "The septohippocampal system: Feedback regulation of cholinergic modulation." Hasselmo presented a model in which feedback regulation sets appropriate dynamics for learning of novel input or recall of familiar input. This model extends previous work on cholinergic modulation of the piriform cortex (Hasselmo, 1993; Hasselmo, 1994). This model depends upon a comparison in region CA1 between self-organized input from entorhinal cortex and recall of patterns of activity associated with CA3 input. When novel afferent input is presented, the inputs to CA1 do not match, and cholinergic modulation remains high, allowing storage of a new association. For familiar input, the match between input patterns suppresses modulation, allowing recall dynamics dominated by input from CA3.

Michael Recce and Neil Burgess, from England, presented their work on "Using phase coding and wave packets to represent places." They are attempting to model the spatial behavior of rats in terms of the firing of single cells in the hippocampus. A reinforcement signal enables a set of "goal cells" to learn a population vector encoding the direction of the rat from the goal. This is achieved by exploiting the apparent phase-coding of place cell firing, and the presence of head-direction cells. The model shows rapid latent-learning and robust navigation to previously encountered goal locations (Burgess, O'Keefe, & Recce, 1993; Burgess, Recce, & O'Keefe, 1994). Spatial trajectories and cell firing characteristics compare well with experimental data.

Richard Granger, of U.C. Irvine, was originally scheduled to talk on "Distinct biology and computation of entorhinal, dentate, CA3 and CA1." Granger and colleagues have noted that synaptic changes in each component of the hippocampus (i.e., DG, CA3 and CA1) exhibit different time courses, specificities, and reversibility. As such, they suggest that subtypes of memory operate serially, in an

"assembly line" of specialized functions, each of which adds a unique aspect to the processing of memories (Granger et al, 1994).

In other talks, Bruce McNaughton of the University of Arizona discussed models of spatial navigation (McNaughton et al, 1991) and William Levy from the University of Virginia presented a theory of how sparse recurrence of CA3 and several other, less direct feedback systems, leads to an ability to learn and compress sequences (Levy, 1989). Mathew Shapiro, of McGill University, had been scheduled to talk on computing locations and trajectories with simulated hippocampal place fields.

**References**

Burgess N, O'Keefe J & Recce M (1993) Using hippocampal "place cells" for navigation, exploiting phase coding, in: Hanson S J, Giles C L & Cowan J D (eds.) Advances in Neural Information Processing Systems 5. San Mateo, CA: Morgan Kaufmann.

Burgess N, Recce M and O'Keefe J (1994) A model of hippocampal function, Neural Networks, Special Issue on Neurodynamics and Behavior, to be published.

Gluck, M. and Granger, R. (1993). Computational models of the neural bases of learning and memory. Annual Review of Neuroscience. 16, 667-706.

Gluck, M., & Myers, C. (1993). Hippocampal mediation of stimulus representation: A computational theory. Hippocampus, 3, 491-516.

Granger, R., Whitson, J., Larson, J. and Lynch, G. (1994). Non-Hebbian properties of LTP enable high-capacity encoding of temporal sequences. Proc. Nat'l. Acad. Sci., (in press).

Hasselmo, M.E. (1993) Acetylcholine and learning in a cortical associative memory. Neural Computation 5, 32-44.

Hasselmo, M.E. (1994) Runaway synaptic modification in models of cortex: Implications for Alzheimer's disease. Neural Networks, in press.

Levy, W. B (1989) A computational approach to hippocampal function. In: Computational Models of Learning in Simple Neural Systems. (R.D. Hawkins and G.H. Bower, Eds.), New York: Academic Press, pp. 243-305.

McClelland, J. L. (1994/in press). The organization of memory: A parallel distributed processing perspective. Revue Neurologique, Masson, Paris

McNaughton, B., Chen, L., & Markus, E. (1991). "Dead reckoning", landmark learning, and the sense of direction: A neurophysiological and computational hypothesis. Journal of Cognitive Neuroscience, 3(2), 190-202.